# Against Edges: Function Approximation with Multiple Support Maps

**Trevor Darrell and Alex Pentland**
Vision and Modeling Group, The Media Lab
Massachusetts Institute of Technology
E15-388, 20 Ames Street
Cambridge MA, 02139

## Abstract

Networks for reconstructing a sparse or noisy function often use an edge field to segment the function into homogeneous regions, This approach assumes that these regions do not overlap or have disjoint parts, which is often false. For example, images which contain regions split by an occluding object can't be properly reconstructed using this type of network. We have developed a network that overcomes these limitations, using support maps to represent the segmentation of a signal. In our approach, the support of each region in the signal is explicitly represented. Results from an initial implementation demonstrate that this method can reconstruct images and motion sequences which contain complicated occlusion.

## 1  Introduction

The task of efficiently approximating a function is central to the solution of many important problems in perception and cognition. Many vision algorithms, for instance, integrate depth or other scene attributes into a dense map useful for robotic tasks such as grasping and collision avoidance. Similarly, learning and memory are often posed as a problem of generalizing from stored observations to predict future behavior, and are solved by interpolating a surface through the observations in an appropriate abstract space. Many control and planning problems can also be solved by finding an optimal trajectory given certain control points and optimization constraints.

In general, of course, finding solutions to these approximation problems is an ill-posed problem, and no exact answer can be found without the application of some prior knowledge or assumptions. Typically, one assumes the surface to be fit is either locally smooth or has some particular parametric form or basis function description. Many successful systems have been built to solve such problems in the cases where these assumptions are valid. However in a wide range of interesting cases where there is no single global model or universal smoothness constraint, such systems have difficulty. These cases typically involve the approximation or estimation of a heterogeneous function whose typical local structure is known, but which also includes an unknown number of abrupt changes or discontinuities in shape.

## 2   Approximation of Heterogeneous Functions

In order to accurately approximate a heterogeneous function with a minimum number of parameters or interpolation units, it is necessary to divide the function into homogeneous chunks which can be approximated parsimoniously. When there is more than one homogeneous chunk in the signal/function, the data must be segmented so that observations of one object do not intermingle with and corrupt the approximation of another region.

One simple approach is to estimate an edge map to denote the boundaries of homogeneous regions in the function, and then to regularize the function within such boundaries. This method was formalized by Geman and Geman (1984), who developed the "line-process" to insert discontinuities in a regularization network. A regularized solution can be efficiently computed by a neural network, either using discrete computational elements or analog circuitry (Poggio et al. 1985; Terzopoulos 1988). In this context, the line-process can be thought of as an array of switches placed between interpolation nodes (Figure 1a). As the regularization proceeds in this type of network, the switches of the line process open and prevent smoothing across suspected discontinuities. Essentially, these switches are opened when the squared difference between neighboring interpolated values exceeds some threshold (Blake and Zisserman 1987; Geiger and Girosi 1991). In practice a continuation method is used to avoid problems with local minima, and a continuous non-linearity is used in place of a boolean discontinuity. The term "resistive fuse" is often used to describe these connections between interpolation sites (Harris et al. 1990).

## 3   Limitations of Edge-based Segmentation

An edge-based representation assumes that homogeneous chunks of a function are completely connected, and have no disjoint subregions. For the visual reconstruction task, this implies that the projection of an object onto the image plane will always yield a single connected region. While this may be a reasonable assumption for certain classes of synthetic images, it is not valid for realistic natural images which contain occlusion and/or transparent phenomena.

While a human observer can integrate over gaps in a region split by occlusion, the line process will prevent any such smoothing, no matter how close the subregions are in the image plane. When these disjoint regions are small (as when viewing an object through branches or leaves), the interpolated values provided by such a

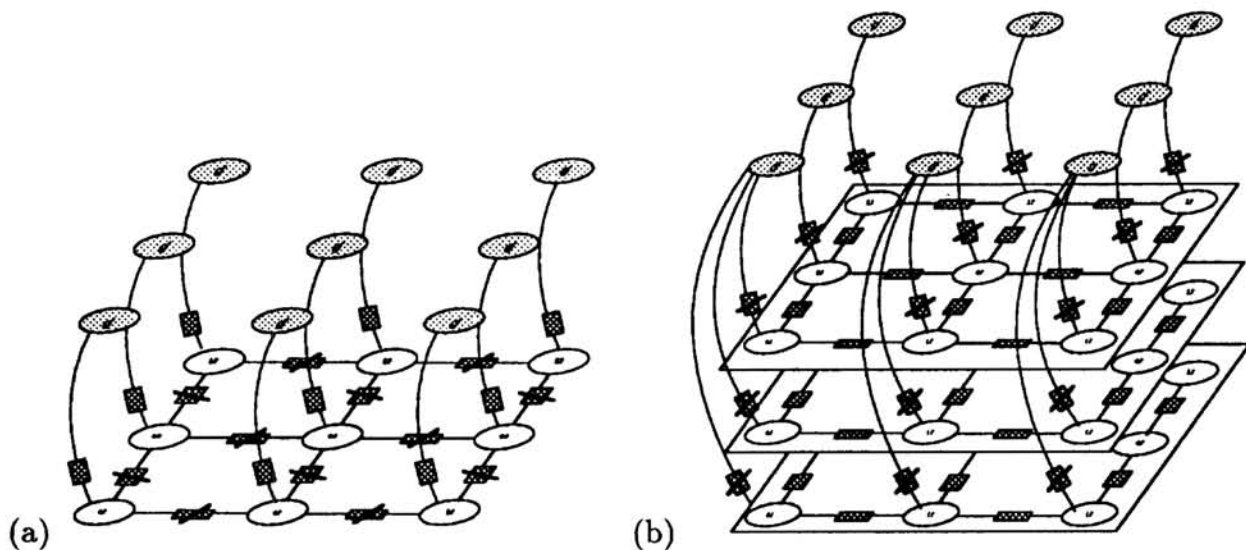

Figure 1: (a) Regularization network with line-process. Shaded circles represent data nodes, while open circles represent interpolation nodes. Solid rectangles indicate resistors; slashed rectangles indicate "resistive fuses". (b) Regularization network with explicit support maps; support process can be implemented by placing resistive fuses between data and interpolation nodes (other constraints on support are described in text).

network will not be reliable, since observation noise can not be averaged over a large number of samples.

Similarly, an edge-based approach cannot account for the perception of motion transparency, since these stimuli have no coherent local neighborhoods. Human observers can easily interpolate 3-D surfaces in transparent random-dot motion displays (Husain et al. 1989). In this type of display, points only last a few frames, and points from different surfaces are transparently intermingled. With a line-process, no smoothing or integration would be possible, since neighboring points in the image belong to different 3-D surfaces. To represent and process images containing this kind of transparent phenomena, we need a framework that does not rely on a global 2D edge map to make segmentation decisions. By generalizing the regularization/surface interpolation paradigm to use support maps rather than a line-process, we can overcome limitations the discontinuity approach has with respect to transparency.

## 4    Using Support Maps for Segmentation

Our approach decomposes a heterogeneous function into a set of individual approximations corresponding to the homogeneous regions of the function. Each approximation covers a specific region, and ues a support map to indicate which points belong to that region. Unlike an edge-based representation, the support of an approximation need not be a connected region — in fact, the support can consist of a scattered collection of independent points!

For a single approximation, it is relatively straight-forward to compute a support map. Given an approximation, we can find the support it has in the function by thresholding the residual error of that approximation. In terms of analog regularization, the support map (or support "process") can be implemented by placing a resistive fuse between the data and the interpolating units (Figure 1b).

A single support map is limited in usefulness, since only one region can be approximated. In fact, it reduces to the "outlier" rejection paradigm of certain robust estimation methods, which are known to have severe theoretical limits on the amount of outlier contamination they can handle (Meer et al. 1991; Li 1985). To represent true heterogeneous stimuli, multiple support maps are needed, with one support map corresponding to each homogeneous (but not necessarily connected) region.

We have developed a method to estimate a set of these support maps, based on finding a minimal length description of the function. We adopt a three-step approach: first, we generate a set of candidate support maps using simple thresholding techniques. Second, we find the subset of these maps which minimally describes the function, using a network optimization to find the smallest set of maps that covers all the observations. Finally, we re-allocate the support in this subset, such that only the approximation with the lowest residual error supports a particular point.

## 4.1   Estimating Initial Support Fields

Ideally, we would like to consider all possible support patterns of a given dimension as candidate support maps. Unfortunately, the combinatorics of the problem makes this impossible; instead, we attempt to find a manageable number of initial maps which will serve as a useful starting point.

A set of candidate approximations can be obtained in many ways. In our work we have initialized their surfaces either using a table of typical values or by fitting a small fixed regions of the function. We denote each approximation of a homogeneous region as a tuple, $(a_i, \vec{s_i}, \vec{u_i}, \vec{r_i})$, where $\vec{s_i} = \{s_{ij}\}$ is a support map, $\vec{u_i} = \{u_{ij}\}$ is the approximated surface, and $\vec{r_i} = \{r_{ij}\}$ is the residual error computed by taking the difference of $\vec{u_i}$ with the observed data. (The scalar $a_i$ is used in deciding which subset of approximations are used in the final representation.) The support fields are set by thresholding the residual field based on our expected (or assumed) observation variance $\theta$.

$$s_{ij} = \left\{ \begin{array}{ll} 1 & if \ (r_{ij})^2 < \theta \\ 0 & otherwise \end{array} \right\}$$

## 4.2   Estimating the Number of Regions

Perhaps the most critical problem in recovering a good heterogeneous description is estimating how many regions are in the function. Our approach to this problem is based on finding a small set of approximations which constitutes a parsimonious description of the function. We attempt to find a subset of the candidate approximations whose support maps are a *minimal covering* of the function, e.g. the smallest subset whose combined support covers the entire function. In non-degenerate cases this will consist of one approximation for each real region in the function.

The quantity $a_i$ indicates if approximation $i$ is included in the final representation. A positive value indicates it is "active" in the representation; a negative value indicates it is excluded from the representation. Initially $a_i$ is set to zero for each approximation; to find a minimal covering, this quantity is dynamically updated as a function of the number of points uniquely supported by a particular support map.

A point is uniquely supported in a support map if it is supported by that map and no other. Essentially, we find these points by modulating the support values of a particular approximation with shunting inhibition from all other active approximations. To compute $c_{ij}$, a flag that indicates whether or not point $j$ of map $i$ is uniquely supported, we multiply each support map with the product of the inverse of all other maps whose $a_i$ value indicates it is active:

$$c_{ij} = s_{ij} \prod_{k \neq i} (1 - s_{kj} \sigma(a_k))$$

where $\sigma()$ is a sigmoid function which converts the real-valued $a_i$ into a multiplicative factor in the range $(0, 1)$. The quantity $c_{ij}$ is close to one at uniquely supported points, and close to zero for all other points.

If there are a sufficient number of uniquely supported points in an approximation, we increase $a_i$, otherwise it is decreased:

$$\frac{d}{dt}a_i = \sum_j c_{ij} - \alpha. \tag{1}$$

where $\alpha$ specifies the penalty for adding another approximation region to the representation. This constant determines the smallest number of points we are willing to have constitute a distinct region in the function. The network defined by these equations has a corresponding Lyoponov function:

$$E = \sum_i^N a_i(-\sum_j^M (\sigma(s_{ij}) \prod_{k \neq i} (1 - \sigma(s_{kj})\sigma(a_k))) + \alpha)$$

so it will be guaranteed to converge to a local minima if we bound the values of $a_i$ (for fixed $s_{ij}$ and $\alpha$). After convergence, those approximations with positive $a_i$ are kept, and the rest are discarded. Empirically we have found the local minima found by our network correspond to perceptually salient segmentations.

## 4.3    Refining Support Fields

Once we have a set of approximations whose support maps minimally cover the function (and presumably correspond to the actual regions of the function), we can refine the support using a more powerful criteria than a local threshold. First, we interpolate the residual error values through unsampled points, so that support can be computed even where there are no observations. Then we update the support maps based on which approximation has the lowest residual error for a given point:

$$s_{ij} = \begin{cases} 1 & if\ (r_{ij})^2 < \theta \\ & and\ (r_{ij})^2 = \min_{\{k|a_k>0\}}(r_{kj})^2 \\ 0 & otherwise \end{cases}$$

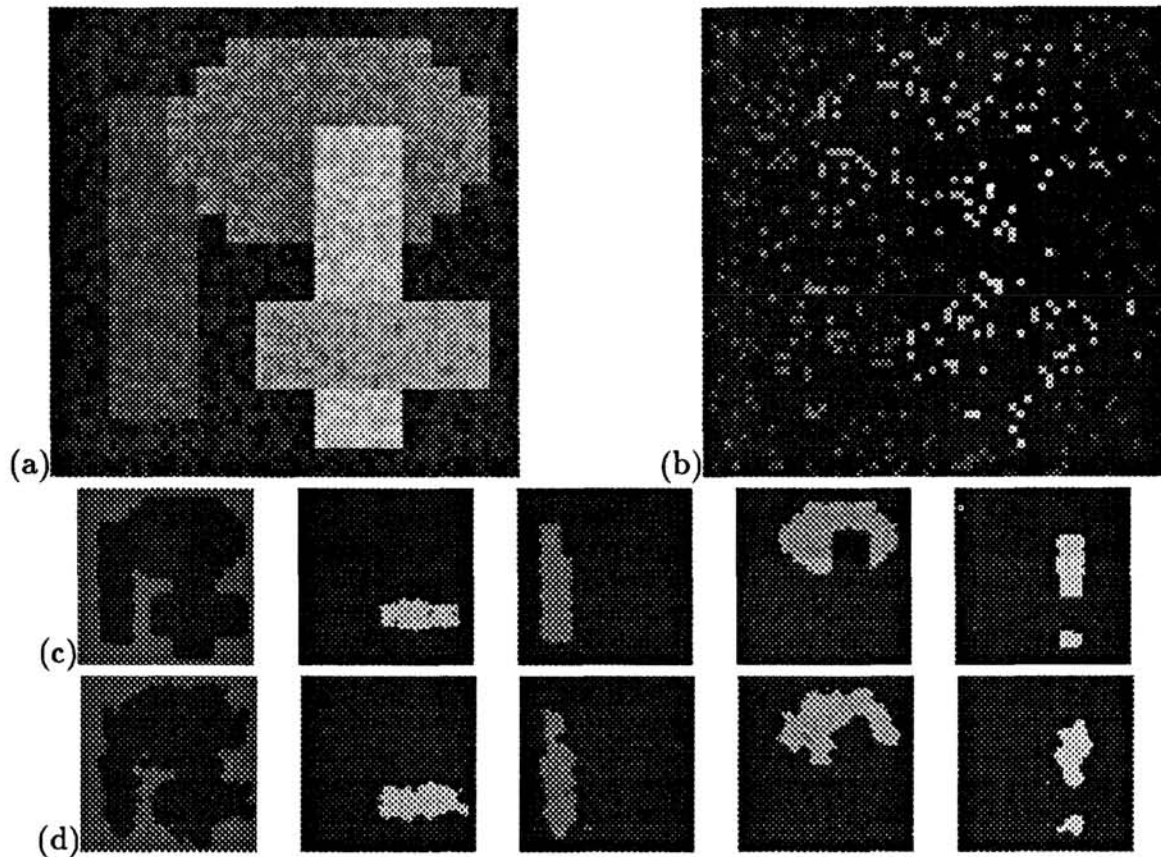

Figure 2: (a) Function consisting of constant regions with added noise. (b) Same function sparsely sampled. (c) Support maps found to approximate uniformly sampled function. (d) Support maps found for sparsely sampled function.

## 5   Results

We tested how well our network could reconstruct functions consisting of piecewise constant patches corrupted with random noise of known variance. Figure 2(a) shows the image containing the function the used in this experiment. We initialized 256 candidate approximations, each with a different constant surface. Since the image consisted of piecewise constant regions, the interpolation performed by each approximation was to compute a weighted average of the data over the supported points. Other experiments have used more powerful shape models, such as thin-plate or membrane Markov random fields, as well as piecewise-quadratic polynomials (Darrell et al. 1990).

Using a penalty term which prevented approximations with 10 or fewer support points to be considered ($\alpha = 10.0$), the network found 5 approximations which covered the entire image; their support maps are shown in Figure 2(c). The estimated surfaces corresponded closely to the values in the constant patches before noise was added. We ran a the same experiment on a sparsely sampled version of this function, as shown in Figure 2(b) and (d), with similar results and only slightly reduced accuracy in the recovered shape of the support maps.

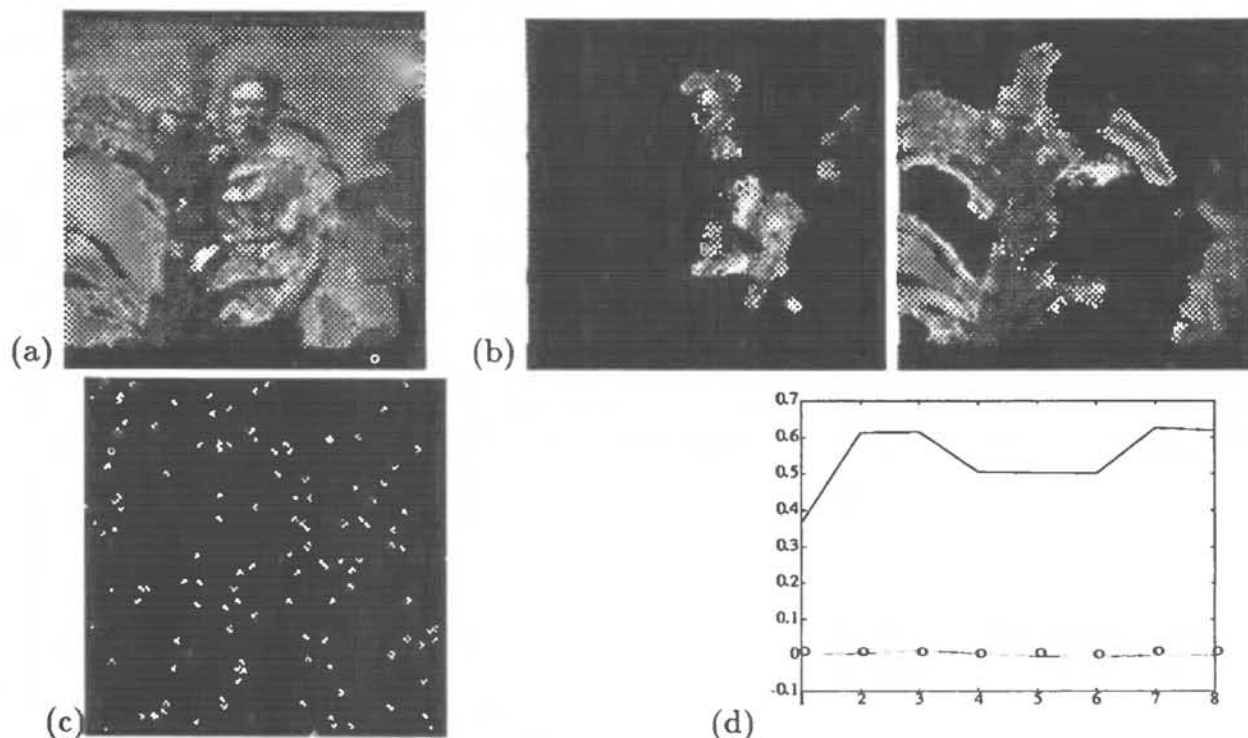

Figure 3: (a) First frame from image sequence and (b) recovered regions. (c) First frame from random dot sequence described in text. (d) Recovered parameter values across frames for dots undergoing looming motion; solid line plots $T_z$, dotted line plots $T_x$, and circles plot $T_y$ for each frame.

We have also applied our framework to the problem of motion segmentation. For homogeneous data, a simple "direct" method can be used to model image motion (Horn and Weldon 1988). Under this assumption, the image intensities for a region centered at the origin undergoing a translation $(T_x, T_y, T_z)$ satisfy at each point

$$0 = \frac{dI}{dt} + T_x \frac{dI}{dx} + T_y \frac{dI}{dy} + T_z (x \frac{dI}{dx} + y \frac{dI}{dy})$$

where $I$ is the image function. Each approximation computes a motion estimate by selecting a $T$ vector which minimizes the square of the right hand side of this equation over its support map, using a weighted least-squares algorithm. The residual error at each point is then simply this constraint equation evaluated with the particular translation estimate.

Figure 3(a) shows the first frame of one sequence, containing a person moving behind a stationary plant. Our network began with 64 candidate approximations, with the initial motion parameters in each distributed uniformly along the parameter axes. Figure 3(b) shows the segmentation provided by our method. Two regions were found to be needed, one for the person and one for the plant. Most of the person has been correctly grouped together despite the occlusion caused by the plant's leaves. Points that have no spatial or temporal variation in the image sequence are not attributed to any approximation, since they are invisible to our motion model. Note that there is a cast shadow moving in synchrony with the person in the scene, and is thus grouped with that approximation.

Finally, we ran our system on the finite-lifetime, transparent random dot stimulus described in Section 2. Since our approach recovers a global motion estimate for each region in each frame, we do not need to build explicit pixel-to-pixel correspondences over long sequences. We used two populations of random dots, one undergoing a looming motion and one a rightward shift. After each frame 10% of the dots died off and randomly moved to a new point on the 3-D surface. Ten 128x128 frames were rendered using perspective projection; the first is shown in Figure 3(c)

We applied our method independently to each trio of successive frames, and in each case two approximations were found to account for the motion information in the scene. Figure 3(d) shows the parameters recovered for the looming motion. Similar results were found for the translating motion, except that the $T_x$ parameter was nonzero rather than $T_z$. Since the recovered estimates were consistent, we would be able to decrease the overall uncertainty by averaging the parameter values over successive frames.

## References

Geman, S., and Geman, D. (1984) Stochastic relaxation, Gibbs distribution, and Bayesian restoration of images. *Trans. Pattern Anal. Machine Intell.* 6:721-741.

Poggio, T., Torre, V., and Koch, C. (1985) Computational vision and regularization theory. *Nature* 317(26).

Terzopoulos, D. (1988) The computation of visible surface representations. *IEEE Trans. Pattern Anal. Machine Intell.* 10:4.

Geiger, D., and Girosi, F. (1991) Parallel and deterministic algorithms from MRF's: surface reconstruction. *Trans. Pattern Anal. Machine Intell.* 13:401-412.

Blake, A. and Zisserman, A. (1987) *Visual Reconstruction*; MIT Press, Cambridge, MA.

Harris J., Koch, C., Staats, E., and Luo, J. (1990) Analog hardware for detecting discontinuities in early vision *Intl. J. Computer Vision* 4:211-233.

Husain, M., Treue, S., and Andersen, R. A. (1989) Surface interpolation in three-dimensional structure-from-motion perception. *Neural Computation* 1:324-333.

Meer, P., Mintz, D., and Rosenfeld, A. (1991) Robust regression methods for computer vision: A review. *Intl. J. Computer Vision*; 6:60-70.

Li, G. (1985) Robust Regression. In D.C. Hoaglin, F. Mosteller and J.W. Tukey (Eds.) *Exploring Data, Tables, Trends and Shapes*: John Wiley & Sons, N.Y.

Darrell, T., Sclaroff, S., and Pentland, A. P. (1990) Segmentation by minimal description. *Proc. IEEE 3nd Intl. Conf. Computer Vision*; Osaka, Japan.

Horn, B.K.P., and Weldon, E.J. (1988) Direct methods for recovering motion. *Intl. J. Computer Vision* 2:51-76.
